# Data clustering by Markovian relaxation and the Information Bottleneck Method

**Naftali Tishby**    and    **Noam Slonim**
School of Computer Science and Engineering and Center for Neural Computation *
The Hebrew University, Jerusalem, 91904 Israel
email: {tishby,noamm}@cs.huji.ac.il

## Abstract

We introduce a new, non-parametric and principled, distance based clustering method. This method combines a pairwise based approach with a vector-quantization method which provide a meaningful interpretation to the resulting clusters. The idea is based on turning the distance matrix into a Markov process and then examine the decay of mutual-information during the relaxation of this process. The clusters emerge as quasi-stable structures during this relaxation, and then are extracted using the information bottleneck method. These clusters capture the information about the initial point of the relaxation in the most effective way. The method can cluster data with no geometric or other bias and makes no assumption about the underlying distribution.

## 1  Introduction

Data clustering is one of the most fundamental pattern recognition problems, with numerous algorithms and applications. Yet, the problem itself is ill-defined: the goal is to find a "reasonable" partition of data points into classes or clusters. What is meant by "reasonable" depends on the application, the representation of the data, and the assumptions about the origins of the data points, among other things.

One important class of clustering methods is for cases where the data is given as a matrix of pairwise distances or (dis)similarity measures. Often these distances come from empirical measurement or some complex process, and there is no direct access, or even precise definition, of the distance function. In many cases this distance does not form a metric, or it may even be non-symmetric. Such data does not necessarily come as a sample of some meaningful distribution and even the issue of generalization and sample to sample fluctuations is not well defined. Algorithms that use only the pairwise distances, without explicit use of the distance measure itself, employ statistical mechanics analogies [3] or collective graph theoretical properties [6], etc. The points are then grouped based on some global criteria, such as connected components, small cuts, or minimum alignment energy. Such algorithms are sometimes computationally inefficient and in most cases it is difficult to interpret the resulting

*Work supported in part by the US-Israel binational science foundation (BSF) and by the Human Frontier Science Project (HFSP). NS is supported by the Levi Eshkol grant.

clusters. I.e., it is hard to determine a common property to all the points in one cluster - other than that the clusters "look reasonable".

A second class of clustering methods is represented by the generalized vector quantization (VQ) algorithm. Here one fits a model (e.g. Gaussian distributions) to the points in each cluster, such that an average (known) distortion between the data points and their corresponding representative is minimized. This type of algorithms may rely on theoretical frameworks, such as rate distortion theory, and provide much better interpretation for the resulting clusters. VQ type algorithms can also be more computationally efficient since they require the calculation of distances, or distortion, between the data and the centroid models only, not between every pair of data points. On the other hand, they require the knowledge of the distortion function and thus make specific assumptions about the underlying structure or model of the data.

In this paper we present a new, information theoretic combination of pairwise clustering with meaningful and intuitive interpretation for the resulting clusters. In addition, our algorithm provides a clear and objective figure of merit for the clusters - without making any assumption on the origin or structure of the data points.

## 2 Pairwise distances and Markovian relaxation

The first step of our algorithm is to turn the pairwise distance matrix into a Markov process, through the following simple intuition. Assign a state of a Markov chain to each of the data points and transition probabilities between the states/points as a function of their pairwise distances. Thus the data can be considered as a directed graph with the points as nodes and the pairwise distances, which need not be symmetric or form a metric, on the arcs of the graph. Distances are normally considered additive, i.e., the length of a trajectory on the graph is the sum of the arc-lengths. Probabilities, on the other hand, are multiplicative for independent events, so if we want the probability of a (random) trajectory on the graph to be naturally related to its length, the transition probabilities between points should be exponential in their distance. Denoting by $d(\mathbf{x}_i, \mathbf{x}_j)$ the pairwise distance between the points $\mathbf{x}_i$ and $\mathbf{x}_j$,[1] then the transition probability that our Markov chain move from the point $\mathbf{x}_j$ at time $t$ to the point $\mathbf{x}_i$ at time $t+1$, $P_{i,j} \equiv p(\mathbf{x}_i(t+1)|\mathbf{x}_j(t))$, is chosen as,

$$p(\mathbf{x}_i(t+1)|\mathbf{x}_j(t)) \propto \exp(-\lambda d(\mathbf{x}_i, \mathbf{x}_j)) , \qquad (1)$$

where $\lambda^{-1}$ is a length scaling factor that equals the mean pairwise distance of the $k$ nearest neighbors to the point $\mathbf{x}_i$. The details of this rescaling are not so important for the final results, and a similar exponentiation of the distances, without our probabilistic interpretation, was performed in other clustering works (see e.g. [3, 6]). A proper normalization of each row is required to turn this matrix into a stochastic transition matrix.

Given this transition matrix, one can imagine a random walk starting at every point on the graph. Specifically, the probability distribution of the positions of a random walk, starting at $\mathbf{x}_j$ after $t$ time steps, is given by the $j$-th row of the $t-th$ iteration of the 1-step transition matrix. Denoting by $P^t$ the $t$-step transition matrix, $P^t = (P)^t$, is indeed the $t$-th power of the 1-step transition probability matrix. The probability of a random walk starting at $\mathbf{x}_j$ at time 0, to be at $\mathbf{x}_i$ at time $t$ is thus,

$$p(\mathbf{x}_i(t)|\mathbf{x}_j(0)) = P^t_{i,j} . \qquad (2)$$

If we assume that all the given pairwise distances are finite we obtain in this way an ergodic Markov process with a single stationary distribution, denoted by $\pi$. This distribution is a right-eigenvector of the t-step transition matrix (for every t), since, $\pi_i = \sum_j P_{i,j} \pi_j$ . It is also the limit distribution of $p(\mathbf{x}_i(t)|\mathbf{x}_j(0))$ for all $j$, i.e., $\lim_{t \to \infty} p(\mathbf{x}_i(t)|\mathbf{x}_j(0)) = \pi_i$. During the dynamics of the Markov process any initial state distribution is going to relax to this final stationary distribution and the information about the initial point of a random walk is completely lost.

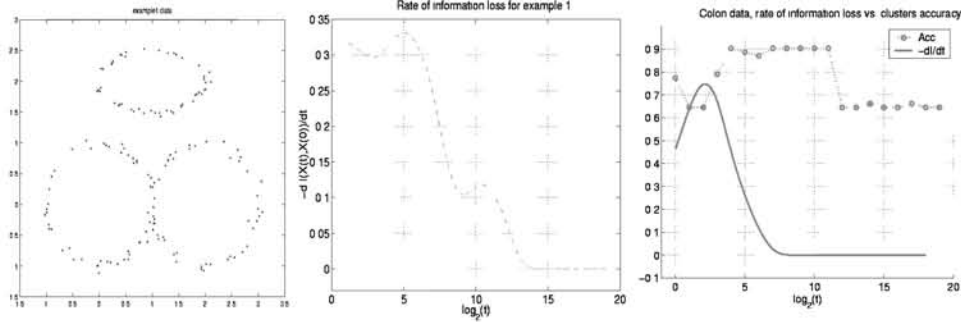

Figure 1: On the left shown an example of data, consisting of 150 points in 2D. On the middle, we plot the rate of information loss, $-\frac{dI(t)}{dt}$, during the relaxation. Notice that the algorithm has no prior information about circles or ellipses. The rate of the information loss is slow when the "random walks" stabilize on some sub structures of the data - our proposed clusters. On the right we plot the rate of information loss for the *colon cancer data*, and the accuracy of the obtained clusters for different relaxation times, with the original classes.

## 2.1 Relaxation of the mutual information

The natural way to quantify the information loss during this relaxation process is by the mutual information between the initial point variable, $X(0) = \{\mathbf{x}_j(0)\}$ and the point of the random walk at time $t$, $X(t) = \{\mathbf{x}_i(t)\}$. The mutual information between the random variables $X$ and $Y$ is the symmetric functional of their joint distribution,

$$I(X;Y) = \sum_{x \in X, y \in Y} p(x,y) \log \left( \frac{p(x,y)}{p(x)p(y)} \right) = \sum_{x \in X, y \in Y} p(x)p(y|x) \log \left( \frac{p(y|x)}{p(y)} \right) .$$
(3)

For the Markov relaxation this mutual information is given by,

$$I(t) \equiv I(X(0); X(t)) = \sum_j p_j \sum_i P_{i,j}^t \log \frac{P_{i,j}^t}{p_i^t} = \sum_j p_j D_{KL}[P_{i,j}^t \| p_i^t] ,$$
(4)

where $p_j$ is the prior probability of the states, and $p_i^t = \sum_j p_{i,j}^t p_j$ is the unconditioned probability of $\mathbf{x}_i$ at time $t$. The $D_{KL}$ is the Kulback-Liebler divergence [4], defined as: $D_{KL}[p\|q] \equiv \sum_y p(y) \log \frac{p(y)}{q(y)}$ which is the information theoretic measure of similarity of distributions. Since all the rows $p_{i,j}^t$ relax to $\pi$ this divergence goes to zero as $t \to \infty$. While it is clear that the information about the initial point, $I(t)$, decays monotonically (exponentially asymptotically) to zero, the *rate* of this decay at finite $t$ conveys much information on the structure of the data points.

Consider, as a simple example, the planer data points shown in figure 1, with $d(\mathbf{x}_i, \mathbf{x}_j) = (x_i - x_j)^2 + (y_i - y_j)^2$. As can be seen, the rate of information loss

about the initial point of the random walk, $-\frac{dI(t)}{dt}$, while always positive - slows down at specific times during the relaxation. These relaxation locations indicate the formation of quasi-stable structures on the graph. At these relaxation times the transition probability matrix is approximately a projection matrix (satisfying $P^{2t} \approx P^t$) where the almost invariant subgraphs correspond to the clusters. These approximate stationary transitions correspond to slow information loss, which can be identified by derivatives of the information loss at time $t$. Another way to see this phenomena is by observing the rows of $P^t$, which are the conditional distributions $p(\mathbf{x}_i(t)|\mathbf{x}_j(0))$. The rows that are almost indistinguishable, following the partial relaxation, correspond to points $\mathbf{x}_j$ with similar conditional distribution on the rest of the graph at time $t$. Such points should belong to the same structure, or cluster on the graph. This can be seen directly by observing the matrix $P^t$ during the relaxation, as shown in figure 2. The quasi-stable structures on the graph, during

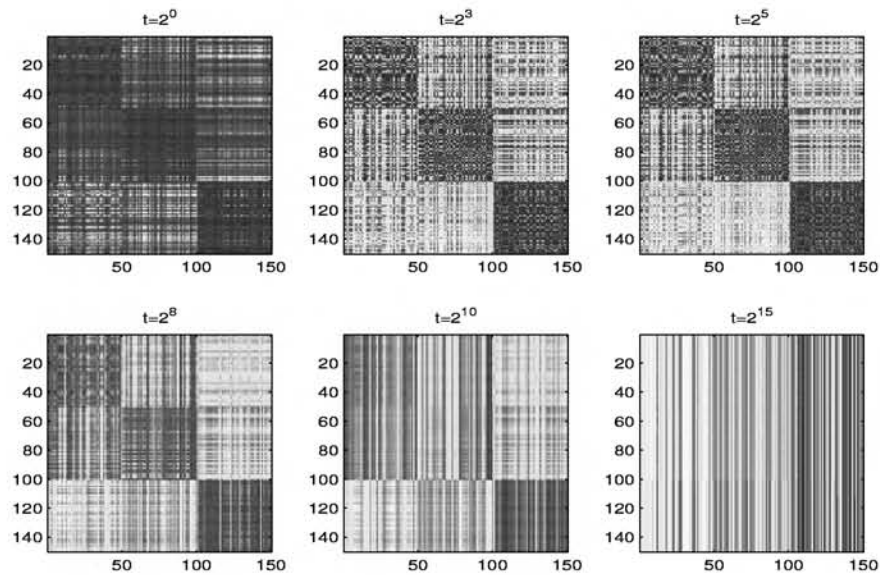

Figure 2: The relaxation process as seen directly on the matrix $P^t$, for different times, for the example data of figure 1. The darker colors correspond to higher probability density in every row. Since the points are ordered by the 3 ellipses, 50 in each ellipse, it is easy to see the clear emergence of 3 blocks of conditional distributions - the rows of the matrix - during the relaxation process. For very large $t$ there is complete relaxation and all the rows equal the stationary distribution of the process. The best correlation between the resulting clusters and the original ellipses (i.e., highest "accuracy" value) is obtained for intermediate times, where the underlying structure emerges.

the relaxation process, are precisely the desirable *meaningful* clusters.

The remaining question pertains to the correct way to group the initial points into clusters that capture the information about the position on the graph after t-steps. In other words, can we replace the initial point with an initial cluster, that enables prediction of the location on the graph at time $t$, with similar accuracy? The answer to this question is naturally provided via the recently introduced *information bottleneck method* [12, 11].

# 3 Clusters that preserve information

The problem of self-organization of the members of a set $X$ based on the similarity of the conditional distributions of the members of another set, $Y$, $\{p(y|x)\}$, was first introduced in [9] and was termed "distributional clustering".

This question was recently shown in [12] to be a specific case of a much more fundamental problem: *What are the features of the variable $X$ that are relevant to the prediction of another, relevance, variable $Y$?* This general problem was shown to have a natural information theoretic formulation: *Find a compressed representation of the variable $X$, denoted $\tilde{X}$, such that the mutual information between $\tilde{X}$ and $Y$, $I(\tilde{X};Y)$, is as high as possible, under a constraint on the mutual information between $X$ and $\tilde{X}$, $I(\tilde{X};X)$.* Surprisingly, this variational principle yields an exact formal solution for the conditional distributions $p(y|\tilde{x})$, $p(\tilde{x}|x)$, and $p(\tilde{x})$. This constrained information optimization problem was called in [12] *The Information Bottleneck Method.*

The original approach to the solution of the resulting equations, used already in [9], was based on an analogy with the "deterministic annealing" (DA) approach to clustering (see [10, 8]). This is a top-down hierarchical algorithm that starts from a single cluster and undergoes a cascade of cluster splits which are determined stochastically (as phase transitions) into a "soft" (fuzzy) tree of clusters. We proposed an alternative approach, based on a greedy bottom-up merging, the "Agglomerative Information Bottleneck" (AIB, see [11]), which is simpler and works better than the DA approach in many situations. This algorithm was applied also in the examples given here.

## 3.1 The information bottleneck method

Given any two non-independent random variables, $X$ and $Y$, the objective of the information bottleneck method is to extract a compact representation of the variable $X$, denoted here by $\tilde{X}$, with minimal loss of mutual information to another, *relevance*, variable $Y$. More specifically, we want to find a (possibly stochastic) map, $p(\tilde{x}|x)$, that maximizes the mutual information to the relevance variable $I(\tilde{X};Y)$, under a constraint on the (lossy) coding length of $X$ via $\tilde{X}$, $I(\tilde{X};X)$. In other words, we want to find an efficient representation of the variable $X$, $\tilde{X}$, such that the predictions of $Y$ from $X$ through $\tilde{X}$ will be as close as possible to the direct prediction of $Y$ from $X$. As shown in [12], by introducing a positive Lagrange multiplier $\beta$ to enforce the mutual information constraint, the problem amounts to maximization of the Lagrangian:

$$\mathcal{L}[p(\tilde{x}|x)] = I(\tilde{X};Y) - \beta^{-1} I(\tilde{X};X) \ , \tag{5}$$

with respect to $p(\tilde{x}|x)$, subject to the Markov condition $\tilde{X} \to X \to Y$ and normalization. This minimization yields directly the following (self-consistent) equations for the map $p(\tilde{x}|x)$, and for $p(y|\tilde{x})$ and $p(\tilde{x})$:

$$\begin{cases} p(\tilde{x}|x) = \frac{p(\tilde{x})}{Z(\beta,x)} \exp\left(-\beta D_{KL}[p(y|x)\|p(y|\tilde{x})]\right) \\ p(y|\tilde{x}) = \sum_x p(y|x)p(\tilde{x}|x)\frac{p(x)}{p(\tilde{x})} \\ p(\tilde{x}) = \sum_x p(\tilde{x}|x)p(x) \end{cases} \tag{6}$$

where $Z(\beta,x)$ is a normalization function. The familiar Kulback-Liebler divergence, $D_{KL}[p(y|x)\|p(y|\tilde{x})]$, *emerges* here from the variational principle. These equations can be solved by iterations that are proved to converge for any finite value of $\beta$

(see [12]). The Lagrange multiplier $\beta$ has the natural interpretation of inverse temperature, which suggests deterministic annealing to explore the hierarchy of solutions in $\tilde{X}$. The variational principle, Eq.(5), determines also the shape of the annealing process, since by changing $\beta$ the mutual informations $I_X \equiv I(\tilde{X}; X)$ and $I_Y \equiv I(\tilde{X}; Y)$ vary such that

$$\frac{\delta I_Y}{\delta I_X} = \beta^{-1} \ . \tag{7}$$

Thus the optimal curve, which is analogous to the rate distortion function in information theory [4], follows a strictly concave curve in the $(I_X, I_Y)$ plane.

The information bottleneck algorithms provide an information theoretic mechanism for identifying the quasi-stable structures on the graph that form our meaningful clusters. In our clustering application the variables are taken as $X = X(0)$ and $Y = X(t)$ during the relaxation process.

## 4 Discussion

When varying the temperature $T = \beta^{-1}$, the information bottleneck algorithms explore the structure of the data in various resolutions. For very low $T$, the resolution is high and each point appears in a cluster of its own. For very high $T$ all points are grouped into one cluster. This process resembles the appearance of the structure during the relaxation. However, there is an important difference between these two mechanisms.

In the bottleneck algorithms clusters are formed by *isotropically blurring* the conditional distributions that correspond to each data point. Points are clustered together when these distributions become sufficiently similar. This process is not sensitive to the global topology of the graph representing the data. This can be understood by looking at the example of figure 1. If we consider two diametrically opposing points on one of the ellipses, they will be clustered together only when their blurred distributions overlap. In this example, unfortunately, this happens when the three ellipses are completely indistinguishable. A direct application of the bottleneck to the original transition matrix is therefore bound to fail in this case.

In the relaxation process, on the other hand, the distributions are merged through the Markovian dynamics on the graph. In our specific example, two opposing points become similar when they reach the other states with similar probabilities following partial relaxation. This process better preserves the fine structure of the underlying graph, and thus enables finer partitioning of the data.

It is thus necessary to combine the two processes. In the first stage, one should relax the Markov process to a quasi-stable point in terms of the rate of information loss. At this point some natural underlying structure emerges, and reflected in the partially relaxed transition matrix, $P^t$. In the second stage we use the information bottleneck algorithm to identify the information preserving clusters.

## 5 More examples

We applied our method to several 'standard' clustering problems and obtained very good results. The first one was the famous "iris data" [7], on which we easily obtained just 5 misclassified points.

A more interesting application was obtained on well known gene expression data, the *Colon cancer data set* provided by Alon et. al [1].This data set consists of

62 tissue samples out of which 22 came from tumors and the rest from "normal" biopsies of colon parts of the same patients. Gene expression levels were given for 2000 genes (oligonucleotides), resulting with a 62 over 2000 matrix.

As done in other studies of this data, we calculated the *Pearson correlation*, $K_p(u,v)$ (see, e.g., [5]), between the $u$ and $v$ expression rows and then transforemed this measure to distances through the simple transformation defined by $d(u,v) = \frac{1-K_p(u,v)}{1+K_p(u,v)}$.

In figure 1 (right panel) we present the rate of information loss for this data and the accuract of the obtained clusters with the original tissue classes. The emrgence of two clusters at the times of "slow" information loss is clearly seen for $t = 2^4$ to $2^{12}$ iterations. The information bottleneck algorithm, when applied at these relaxation times, discovers the original tissue classes, up to 6 or 7 "misclassified" tissues (see figure). For comparison, seven more sophisticated *supervised*, techniques applied in [2] to this data. Six of them had 12 misclassified points or more, and their best results had 7 missclasifed tissues.

## Footnotes

[1]Henceforth we take the number of data points to be $n$ and the point indices run implicitly from 1 to $n$ unless stated otherwise.

# References

[1] U. Alon, N. Barkai, D.A. Notterman, K. Gish, D. Mack, and A.J. Levine. Broad patterns of gene expression revealed by clustering analysis of tumor and normal colon tissues probed by oligonucleotide arrays. Proc. Nat. Acad. Sci. USA, 96:6745-6750, 1999.

[2] A. Ben-Dor, L. Bruhn, N. Friedman, I. Nachman, M. Schummer, and Z. Yakhini. Tissue Classification with Gene Expression Profiles. Journal of Computational Biology, 2000, to appear.

[3] M. Blatt, M. Wiesman, and E. Domany. Data clustering using a model granular magnet. Neural Computation 9, 1805-1842, 1997.

[4] T. M. Cover and J. A. Thomas. *Elements of Information Theory.* John Wiley & Sons, New York, 1991.

[5] M. Eisen, P. Spellman, P. Brown and D. Botstein. Cluster analysis and display of genome wide expression patterns. Proc. Nat. Acad. Sci. USA 95, 14863-14868, 1998.

[6] Y. Gdalyahu, D. Weinshall, and M. Werman, Randomized algorithm for pairwise clustering. in proceedings of NIPS-11, 424–430, 1998.

[7] R.A. Fisher. The use of multiple measurements in taxonomic problems Annual Eugenics, 7, Part II, 179-188, 1936.

[8] T. Hofmann and J. M. Buhmann. Pairwise data clustering by deterministic annealing. *IEEE Transactions on PAMI*, 19(1):1–14, 1997.

[9] F.C. Pereira, N. Tishby, and L. Lee. Distributional clustering of English words. In *30th Annual Meeting of the Association for Computational Linguistics, Columbus, Ohio*, pages 183–190, 1993.

[10] K. Rose. Deterministic Annealing for Clustering, Compression, Classification, Regression, and Related Optimization Problems. *Proceedings of the IEEE*, 86(11):2210–2239, 1998.

[11] N. Slonim and N. Tishby. Agglomerative Information Bottleneck. in proceedings of NIPS-12, 1999.

[12] N. Tishby, F.C. Pereira, and W. Bialek. The information bottleneck method. In proceedings of the 37-th Annual Allerton Conference on Communication, Control and Computing, 368–377, 1999.
